# Connectionist Music Composition Based on Melodic and Stylistic Constraints

**Michael C. Mozer**
Department of Computer Science
and Institute of Cognitive Science
University of Colorado
Boulder, CO 80309-0430

**Todd Soukup**
Department of Electrical
and Computer Engineering
University of Colorado
Boulder, CO 80309-0425

## Abstract

We describe a recurrent connectionist network, called CONCERT, that uses a set of melodies written in a given style to compose new melodies in that style. CONCERT is an extension of a traditional algorithmic composition technique in which transition tables specify the probability of the next note as a function of previous context. A central ingredient of CONCERT is the use of a psychologically-grounded representation of pitch.

## 1 INTRODUCTION

In creating music, composers bring to bear a wealth of knowledge about musical conventions. If we hope to build automatic music composition systems that can mimic the abilities of human composers, it will be necessary to incorporate knowledge about musical conventions into the systems. However, this knowledge is difficult to express: even human composers are unaware of many of the constraints under which they operate.

In this paper, we describe a connectionist network that composes melodies. The network is called CONCERT, an acronym for connectionist composer of erudite tunes. Musical knowledge is incorporated into CONCERT via two routes. First, CONCERT is trained on a set of sample melodies from which it extracts rules of note and phrase progressions. Second, we have built a representation of pitch into CONCERT that is based on psychological studies of human perception. This representation, and an associated theory of generalization proposed by Shepard (1987), provides CONCERT with a basis for judging the similarity among notes, for selecting a response, and for restricting the set of alternatives that can be considered at any time.

## 2 TRANSITION TABLE APPROACHES TO COMPOSITION

We begin by describing a traditional approach to algorithmic music composition using Markov *transition tables*. This simple but interesting technique involves selecting notes sequentially according to a table that specifies the probability of the next note as a func-

tion of the current note (Dodge & Jerse, 1985). The tables may be hand-constructed according to certain criteria or they may be set up to embody a particular musical style. In the latter case, statistics are collected over a set of examples (hereafter, the *training set*) and the table entries are defined to be the transition probabilities in these examples.

In melodies of any complexity, musical structure cannot be fully described by pairwise statistics. To capture additional structure, the transition table can be generalized from a two-dimensional array to $n$ dimensions. In the $n$-dimensional table, often referred to as a table of order $n-1$, the probability of the next note is indicated as a function of the previous $n-1$ notes. Unfortunately, extending the transition table in this manner gives rise to two problems. First, the size of the table explodes exponentially with the amount of context and rapidly becomes unmanageable. Second, a table representing the high-order structure masks whatever low-order structure is present.

Kohonen (1989) has proposed a scheme by which only the *relevant* high-order structure is represented. The scheme is symbolic algorithm that, given a training set of examples, produces a collection of rules — a context-sensitive grammar — sufficient for reproducing most or all of the structure inherent in the set. However, because the algorithm attempts to produce deterministic rules — rules that always apply in a given context — the algorithm will not discover regularities unless they are absolute; it is not equipped to deal with statistical properties of the data. Both Kohonen's musical grammar and the transition table approach suffer from the further drawback that a symbolic representation of notes does not facilitate generalization. For instance, invariance under transposition is not directly representable. In addition, other similarities are not encoded, for example, the congruity of octaves.

Connectionist learning algorithms offer the potential of overcoming the various limitations of transition table approaches and Kohonen musical grammars. Connectionist algorithms are able to discover relevant structure and statistical regularities in sequences (e.g., Elman, 1990; Mozer, 1989), and to consider varying amounts of context, noncontiguous context, and combinations of low-order and high-order regularities. Connectionist approaches also promise better generalization through the use of distributed representations. In a local representation, where each note is represented by a discrete symbol, the sort of statistical contingencies that can be discovered are among notes. However, in a distributed representation, where each note is represented by a set of continuous feature values, the sort of contingencies that can be discovered are among *features*. To the extent that two notes share features, featural regularities discovered for one note may transfer to the other note.

## 3    THE CONCERT ARCHITECTURE

CONCERT is a recurrent network architecture of the sort studied by Elman (1990). A melody is presented to it, one note at a time, and its task at each point in time is to predict the next note in the melody. Using a training procedure described below, CONCERT's connection strengths are adjusted so that it can perform this task correctly for a set of training examples. Each example consists of a sequence of notes, each note being characterized by a pitch and a duration. The current note in the sequence is represented in the input layer of CONCERT, and the prediction of the next note is represented in the output layer. As Figure 1 indicates, the next note is encoded in two different ways: The next-note-distributed (or *NND*) layer contains CONCERT's internal representation of the

note, while the next-note-local (or *NNL*) layer contains one unit for each alternative. For now, it should suffice to say that the representation of a note in the NND layer, as well as in the input layer, is distributed, i.e., a note is indicated by a *pattern* of activity across the units. Because such patterns of activity can be quite difficult to interpret, the NNL layer provides an alternative, explicit representation of the possibilities.

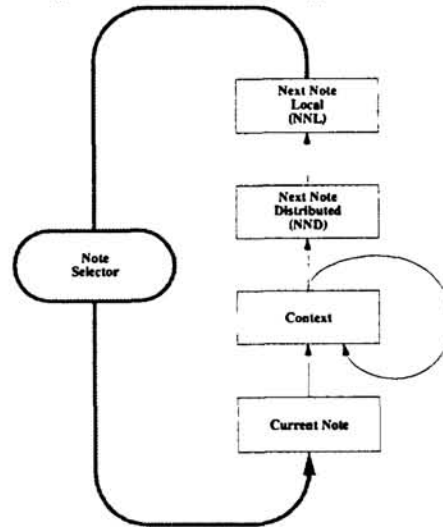

Figure 1: The CONCERT Architecture

The context layer represents the the temporal context in which a prediction is made. When a new note is presented in the input layer, the current context activity pattern is integrated with the new note to form a new context representation. Although CONCERT could readily be wired up to behave as a $k$-th order transition table, the architecture is far more general. The training procedure attempts to determine which aspects of the input sequence are relevant for making future predictions and retain only this task-relevant information in the context layer. This contrasts with Todd's (1989) seminal work on connectionist composition in which the recurrent context connections are prewired and fixed, which makes the nature of the information Todd's model retains independent of the examples on which it is trained.

Once CONCERT has been trained, it can be run in *composition mode* to create new pieces. This involves first seeding CONCERT with a short sequence of notes, perhaps the initial notes of one of the training examples. From this point on, the output of CONCERT can be fed back to the input, allowing CONCERT to continue generating notes without further external input. Generally, the output of CONCERT does not specify a single note with absolute certainty; instead, the output is a probability distribution over the set of candidates. It is thus necessary to select a particular note in accordance with this distribution. This is the role of the selection process depicted in Figure 1.

## 3.1    ACTIVATION RULES AND TRAINING PROCEDURE

The activation rule for the context units is

$$c_i(n) = s\left[\sum_j w_{ij} x_j(n) + \sum_j v_{ij} c_j(n-1)\right],\tag{1}$$

where $c_i(n)$ is the activity of context unit $i$ following processing of input note $n$ (which

we refer to as *step n*), $x_j(n)$ is the activity of input unit $j$ at step $n$, $w_{ij}$ is the connection strength from unit $j$ of the input to unit $i$ of the context layer, and $v_{ij}$ is the connection strength from unit $j$ to unit $i$ within the context layer, and $s$ is a sigmoid activation function rescaled to the range (-1,1). Units in the NND layer follow a similar rule:

$$nnd_i(n) = s\left[\sum_j u_{ij} c_j(n)\right],$$

where $nnd_i(n)$ is the activity of NND unit $i$ at step $n$ and $u_{ij}$ is the strength of connection from context unit $j$ to NND unit $i$.

The transformation from the NND layer to the NNL layer is achieved by first computing the distance between the NND representation, $\mathbf{nnd}(n)$, and the target (distributed) representation of each pitch $i$, $\rho_i$:

$$d_i = |\mathbf{nnd}(n) - \rho_i|,$$

where $|\cdot|$ denotes the L2 vector norm. This distance is an indication of how well the NND representation matches a particular pitch. The activation of the NNL unit corresponding to pitch $i$, $nnl_i$, increases inversely with the distance:

$$nnl_i(n) = e^{-d_i}/\sum_j e^{-d_j}.$$

This normalized exponential transform (proposed by Bridle, 1990, and Rumelhart, in press) produces an activity pattern over the NNL units in which each unit has activity in the range (0,1) and the activity of all units sums to 1. Consequently, the NNL activity pattern can be interpreted as a probability distribution — in this case, the probability that the next note has a particular pitch.

CONCERT is trained using the back propagation unfolding-in-time procedure (Rumelhart, Hinton, & Williams, 1986) using the log likelihood error measure

$$E = -\sum_{p,n} \log nnl_{tgt}(n,p),$$

where $p$ is an index over pieces in the training set and $n$ an index over notes within a piece; *tgt* is the target pitch for note $n$ of piece $p$.

## 3.2   PITCH REPRESENTATION

Having described CONCERT's architecture and training procedure, we turn to the representation of pitch. To accommodate a variety of music, CONCERT needs the ability to represent a range of about four octaves. Using standard musical notation, these pitches are labeled as follows: C1, D1, ..., B1, C2, D2, ... B2, C3, ... C5, where C1 is the lowest pitch and C5 the highest. Sharps are denoted by a #, e.g., F#3. The range C1-C5 spans 49 pitches.

One might argue that the choice of a pitch representation is not critical because back propagation can, in principle, discover an alternative representation well suited to the task. In practice, however, researchers have found that the choice of external representation is a critical determinant of the network's ultimate performance (e.g., Denker et al., 1987; Mozer, 1987). Quite simply, the more task-appropriate information that is built into the network, the easier the job the learning algorithm has. Because we are asking the net-

work to make predictions about melodies that *people* have composed or to generate melodies that *people* perceive as pleasant, we have furnished CONCERT with a psychologically-motivated representation of pitch. By this, we mean that notes that people judge to be similar have similar representations in the network, indicating that the representation in the head matches the representation in the network.

Shepard (1982) has studied the similarity of pitches by asking people to judge the perceived similarity of pairs of pitches. He has proposed a theory of generalization (Shepard, 1987) in which the similarity of two items is exponentially related to their distance in an internal or "psychological" representational space. (This is one justification for the NNL layer computing an exponential function of distance.) Based on psychophysical experiments, he has proposed a five-dimensional space for the representation of pitch, depicted in Figure 2.

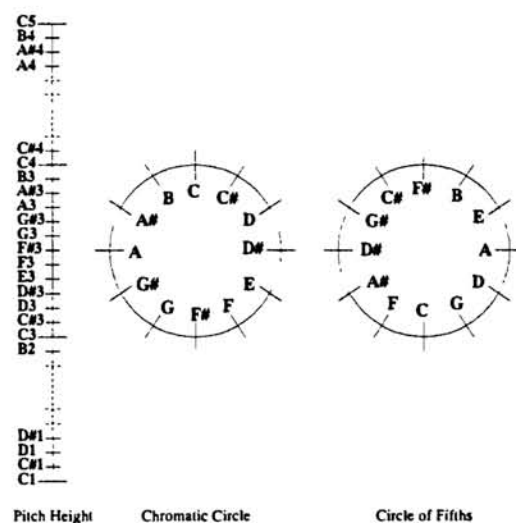

Pitch Height          Chromatic Circle                Circle of Fifths

Figure 2:  Pitch Representation Proposed by Shepard (1982)

In this space, each pitch specifies a point along the *pitch height* (or *PH*) dimension, an $(x,y)$ coordinate on the *chromatic circle* (or *CC*), and an $(x,y)$ coordinate on the *circle of fifths* (or *CF*). we will refer to this representation as PHCCCF, after its three components. The pitch height component specifies the logarithm of the frequency of a pitch; this logarithmic transform places tonal half-steps at equal spacing from one another along the pitch height axis. In the chromatic circle, neighboring pitches are a tonal half-step apart. In the circle of fifths, the perfect fifth of a pitch is the next pitch immediately counterclockwise. Figure 2 shows the relative magnitude of the various components to scale. The proximity of two pitches in the five-dimensional PHCCCF space can be determined simply by computing the Euclidean distance between their representations.

A straightforward scheme for translating the PHCCCF representation into an activity pattern over a set of connectionist units is to use five units, one for pitch height and two pairs to encode the $(x,y)$ coordinates of the pitch on the two circles. Due to several problems, we have represented each circle over a set of 6 binary-valued units that preserves the essential distance relationships among tones on the circles (Mozer, 1990). The PHCCCF representation thus consists of 13 units altogether. Rests (silence) are assigned a code that distinguish them from all pitches. The end of a piece is coded by several rests.

## 4    SIMULATION EXPERIMENTS

### 4.1    LEARNING THE STRUCTURE OF DIATONIC SCALES

In this simulation, we trained CONCERT on a set of diatonic scales in various keys over a one octave range, e.g., D1 E1 F#1 G1 A1 B1 C#2 D2. Thirty-seven such scales can be made using pitches in the C1-C5 range. The training set consisted of 28 scales — roughly 75% of the corpus — selected at random, and the test set consisted of the remaining 9. In 10 replications of the simulation using 20 context units, CONCERT mastered the training set in approximately 55 passes. Generalization performance was tested by presenting the scales in the test set one note at a time and examining CONCERT's prediction. Of the 63 notes to be predicted in the test set, CONCERT achieved remarkable performance: 98.4% correct. The few errors were caused by transposing notes one full octave or one tonal half step.

To compare CONCERT with a transition table approach, we built a second-order transition table from the training set data and measured its performance on the test set. The transition table prediction (i.e., the note with highest probability) was correct only 26.6% of the time. The transition table is somewhat of a straw man in this environment: A transition table that is based on absolute pitches is simply unable to generalize correctly. Even if the transition table encoded relative pitches, a third-order table would be required to master the environment. Kohonen's musical grammar faces the same difficulties as a transition table.

### 4.2    LEARNING INTERSPERSED RANDOM WALK SEQUENCES

The sequences in this simulation were generated by interspersing the elements of two simple random walk sequences. Each interspersed sequence had the following form: $a_1$, $b_1$, $a_2$, $b_2$, $\cdots$, $a_5$, $b_5$, where $a_1$ and $b_1$ are randomly selected pitches, $a_{i+1}$ is one step up or down from $a_i$ on the C major scale, and likewise for $b_{i+1}$ and $b_i$. Each sequence consisted of ten notes. CONCERT, with 25 context units, was trained on 50 passes through a set of 200 examples and was then tested on an additional 100. Because it is impossible to predict the second note in the interspersed sequences ($b_1$) from the first ($a_1$), this prediction was ignored for the purpose of evaluating CONCERT's performance. CONCERT achieved a performance of 91.7% correct. About half the errors were ones in which CONCERT transposed a correct prediction by an octave. Excluding these errors, performance improved to 95.8% correct.

To capture the structure in this environment, a transition table approach would need to consider at least the previous two notes. However, such a transition table is not likely to generalize well because, if it is to be assured of predicting a note at step $n$ correctly, it must observe the note at step $n-2$ in the context of *every possible* note at step $n-1$. We constructed a second-order transition table from CONCERT's training set. Using a testing criterion analogous to that used to evaluate CONCERT, the transition table achieved a performance level on the test set of only 67.1% correct. Kohonen's musical grammar would face the same difficulty as the transition table in this environment.

## 4.3    GENERATING NEW MELODIES IN THE STYLE OF BACH

In a final experiment, we trained CONCERT on the melody line of a set of ten simple minuets and marches by J. S. Bach. The pieces had several voices, but the melody generally appeared in the treble voice. Importantly, to naive listeners the extracted melodies sounded pleasant and coherent without the accompaniment.

In the training data, each piece was terminated with a rest marker (the only rests in the pieces). This allowed CONCERT to learn not only the notes within a piece but also when the end of the piece was reached. Further, each major piece was transposed to the key of C major and each minor piece to the key of A minor. This was done to facilitate learning because the pitch representation does not take into account the notion of musical key; a more sophisticated pitch representation might avoid the necessity of this step.

In this simulation, each note was represented by a duration as well as a pitch. The duration representation consisted of five units and was somewhat analogous the PHCCCF representation for pitch. It allowed for the representation of sixteenth, eighth, quarter, and half notes, as well as triplets. Also included in this simulation were two additional input ones. One indicated whether the piece was in a major versus minor key, the other indicated whether the piece was in 3/4 meter versus 2/4 or 4/4. These inputs were fixed for a given piece.

Learning the examples involves predicting a total of 1,260 notes altogether, no small feat. CONCERT was trained with 40 hidden units for 3000 passes through the training set. The learning rate was gradually lowered from .0004 to .0002. By the completion of training, CONCERT could correctly predict about 95% of the pitches and 95% of the durations correctly. New pieces can be created by presenting a few notes to start and then running CONCERT in composition mode. One example of a composition produced by CONCERT is shown in Figure 3. The primary deficiency of CONCERT's compositions is that they are lacking in global coherence.

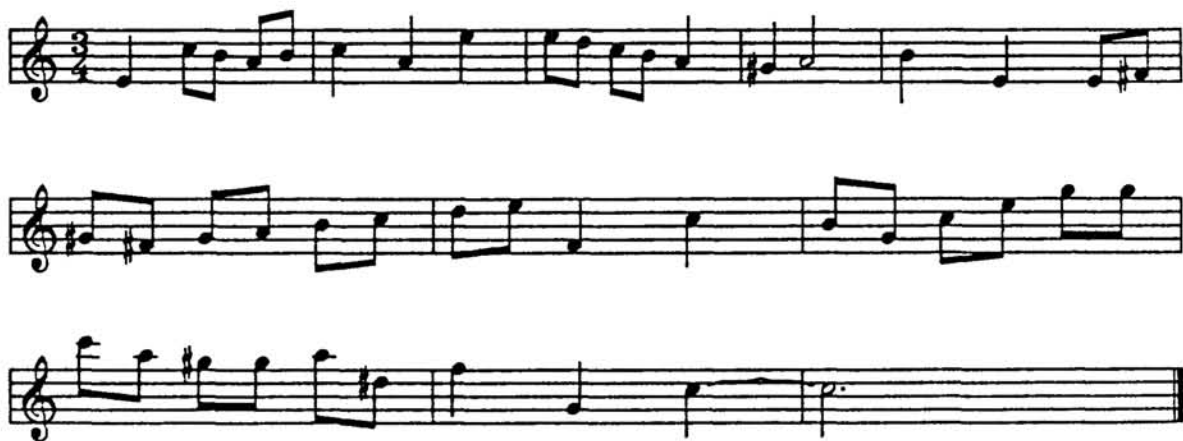

Figure 3:  A Sample Composition Produced by CONCERT

## 5  DISCUSSION

Initial results from CONCERT are encouraging. CONCERT is able to learn musical structure of varying complexity, from random walk sequences to Bach pieces containing nearly 200 notes. We presented two examples of structure that CONCERT can learn but that cannot be captured by a simple transition table or by Kohonen's musical grammar.

Beyond a more systematic examination of alternative architectures, work on CONCERT is heading in two directions. First, the pitch representation is being expanded to account for the perceptual effects of musical context and musical key. Second, CONCERT is being extended to better handle the processing of global structure in music. It is unrealistic to expect that CONCERT, presented with a linear string of notes, could induce not only local relationships among the notes, but also more global phrase structure, e.g., an AABA phrase pattern. To address the issue of global structure, we have designed a network that operates at several different temporal resolutions simultaneously (Mozer, 1990).

### Acknowledgements

This research was supported by NSF grant IRI-9058450, grant 90-21 from the James S. McDonnell Foundation. Our thanks to Paul Smolensky, Yoshiro Miyata, Debbie Breen, and Geoffrey Hinton for helpful comments regarding this work, and to Hal Eden and Darren Hardy for technical assistance.

### References

Bridle, J. (1990). Training stochastic model recognition algorithms as networks can lead to maximum mutual information estimation of parameters. In D. S. Touretzky (Ed.), *Advances in neural information processing systems 2* (pp. 211–217). San Mateo, CA: Morgan Kaufmann.

Dodge, C., & Jerse, T. A. (1985). *Computer music: Synthesis, composition, and performance.* New York: Shirmer Books.

Elman, J. L. (1990). Finding structure in time. *Cognitive Science, 14,* 179–212.

Kohonen, T. (1989). A self-learning musical grammar, or "Associative memory of the second kind". *Proceedings of the 1989 International Joint Conference on Neural Networks,* 1–5.

Mozer, M. C. (1987). RAMBOT: A connectionist expert system that learns by example. In M. Caudill & C. Butler (Eds.), *Proceedings fo the IEEE First Annual International Conference on Neural Networks* (pp. 693–700). San Diego, CA: IEEE Publishing Services.

Mozer, M. C. (1989). A focused back-propagation algorithm for temporal pattern recognition. *Complex Systems, 3,* 349–381.

Mozer, M. C. (1990). *Connectionist music composition based on melodic, stylistic, and psychophysical constraints* (Tech Report CU–CS–495–90). Boulder, CO: University of Colorado, Department of Computer Science.

Rumelhart, D. E., Hinton, G. E., & Williams, R. J. (1986). Learning internal representations by error propagation. In D. E. Rumelhart & J. L. McClelland (Eds.), *Parallel distributed processing: Explorations in the microstructure of cognition. Volume I: Foundations* (pp. 318–362). Cambridge, MA: MIT Press/Bradford Books.

Rumelhart, D. E. (in press). Connectionist processing and learning as statistical inference. In Y. Chauvin & D. E. Rumelhart (Eds.), *Backpropagation: Theory, architectures, and applications.* Hillsdale, NJ: Erlbaum.

Shepard, R. N. (1982). Geometrical approximations to the structure of musical pitch. *Psychological Review, 89,* 305–333.

Shepard, R. N. (1987). Toward a universal law of generalization for psychological science. *Science, 237,* 1317–1323. Shepard (1987)

Todd, P. M. (1989). A connectionist approach to algorithmic composition. *Computer Music Journal, 13,* 27–43.